# A Cortically-Plausible Inverse Problem Solving Method Applied to Recognizing Static and Kinematic 3D Objects

**David W. Arathorn**

Center for Computational Biology,
Montana State University
Bozeman, MT 59717

dwa@cns.montana.edu

General Intelligence Corporation
dwa@giclab.com

## Abstract

Recent neurophysiological evidence suggests the ability to interpret biological motion is facilitated by a neuronal "mirror system" which maps visual inputs to the pre-motor cortex. If the common architecture and circuitry of the cortices is taken to imply a common computation across multiple perceptual and cognitive modalities, this visual-motor interaction might be expected to have a unified computational basis. Two essential tasks underlying such visual-motor cooperation are shown here to be simply expressed and directly solved as transformation-discovery inverse problems: (a) discriminating and determining the pose of a primed 3D object in a real-world scene, and (b) interpreting the 3D configuration of an articulated kinematic object in an image. The recently developed map-seeking method provides a mathematically tractable, cortically-plausible solution to these and a variety of other inverse problems which can be posed as the discovery of a composition of transformations between two patterns. The method relies on an ordering property of superpositions and on decomposition of the transformation spaces inherent in the generating processes of the problem.

## 1 Introduction

A variety of "brain tasks" can be tersely posed as transformation-discovery problems. Vision is replete with such problems, as is limb control. The problem of recognizing the 2D projection of a known 3D object is an inverse problem of finding both the visual and pose transformations relating the image and the 3D model of the object. When the object in the image may be one of many known objects another step is added to the inverse problem, because there are multiple

candidates each of which must be mapped to the input image with possibly different transformations. When the known object is not rigid, the determination of articulations and/or morphings is added to the inverse problem. This includes the general problem of recognition of biological articulation and motion, a task recently attributed to a neuronal mirror-system linking visual and motor cortical areas [1].

Though the aggregate transformation space implicit in such problems is vast, a recently developed method for exploring vast transformation spaces has allowed some significant progress with a simple unified approach. The map-seeking method [2,4] is a general purpose mathematical procedure for finding the decomposition of the aggregate transformation between two patterns, even when that aggregate transformation space is vast and there is no prior information is available to restrict the search space. The problem of concurrently searching a large collection of memories can be treated as a subset of the transformation problem and consequently the same method can be applied to find the best transformation between an input image and a collection of memories (numbering at least thousands in practice to date) during a single convergence. In the last several years the map-seeking method has been applied to a variety of practical problems, most of them related to vision, a few related to kinematics, and some which do not correspond to usual categories of "brain functions." The generality of the method is due to the fact that only the mappings are specialized to the task. The mathematics of the search, whether expressed in an algorithm or in a neuronal or electronic circuit, do not change. From an evolutionary biological point of view this is a satisfying characteristic for a model of cortical function because only the connectivity which implements the mappings must be varied to specialize a cortex to a task. All the rest – organization and dynamics – would remain the same across cortical areas.

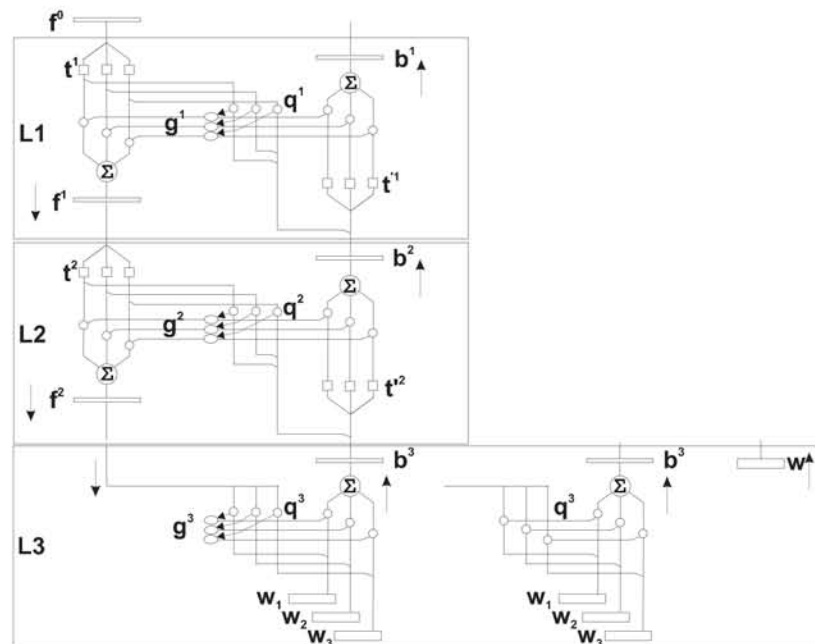

Figure 1. Data flow in map-seeking circuit

Cortical neuroanatomy offers emphatic hints about the characteristics of its solution in the vast neuronal resources allocated to creating reciprocal top-down and bottom-up pathways. More specifically, recent evidence suggests this reciprocal pathway architecture appears to be organized with reciprocal, co-centered fan outs in the opposing directions [3], quite possibly implementing inverse mappings. The data flow of map-seeking computations, seen in Figure 1, is architecturally compatibility with these features of cortical organization. Though not within the scope of this discussion, it has been demonstrated [4] that the mathematical expression of the map-seeking method, seen in equations 6-9 below, has an isomorphic implementation in neuronal circuitry with reasonably realistic dendritic architecture and dynamics (e.g. compatible with [5] ) and oscillatory dynamics.

## 2   The basis for tractable transformation-discovery

The related problems of recognition/interpretation of 2D images of static and articulated kinematic 3D objects illustrate how cleanly significant vision problems may be posed and solved as transformation-discovery inverse problems. The visual and pose (in the sense of orientation) transformations, $t^{visual}$ and $t^{pose}$, between a given 3D model $m_1$ and the extent of an input image containing a 2D projection $P(o_1)$ of an object $o_1$ mappable to $m_1$ can be expressed

$$P(o_1) = t_j^{visual} \circ t_k^{pose}(m_1) \qquad t_j^{visual} \in T^{visual}, t_k^{pose} \in T^{pose} \qquad \text{eq. 1}$$

If we now consider that the model $m_1$ may be constructed by the one-to-many mapping of a base vector or feature $\mathbf{e}$, and that arbitrarily other models $m_i$ may be similarly constructed by different mappings, then the transformation $t^{formation}$ corresponding to the correct "memory" converts the memory database search problem into another transformation-discovery problem with one more composed transformation[1]

$$P(o_1) = t_j^{visual} \circ t_k^{pose} \circ t_{m_1}^{formation}(\mathbf{e}) \qquad t_{m_1}^{formation} \in T^{formation}$$
$$t_{m_1}^{formation}(\mathbf{e}) = m_l \quad m_l \in M \qquad \text{eq. 2}$$

Finally, if we allow a morphable object to be "constructed" by a generative model, whose various configurations or articulations may be generated by a composition of transformations $t^{generative}$ of some root or seed feature $\mathbf{e}$, the problem of explicitly recognizing the particular configuration of morph becomes a transformation-discovery problem of the form

$$P(C_l(o)) = t_j^{visual} \circ t_k^{pose} \circ t_l^{generative}(\mathbf{e}) \qquad t_l^{generative} \in T^{generative} \qquad \text{eq. 3}$$

These unifying formulations are only useful, however, if there is a tractable method of solving for the various transformations. That is what the map-seeking method provides. Abstractly the problem is the discovery of a composition of transformations between two patterns. In general the transformations express the generating process of the problem. Define correspondence $c$ between vectors $\mathbf{r}$ and $\mathbf{w}$ through a composition of $L$ transformations $t_{j1}^1, t_{j2}^2, \cdots, t_{jL}^L$ where $t_{jl}^l \in t_1^l, t_2^l, \cdots, t_{nl}^l$

$$c(\mathbf{j}) = \left\langle \mathop{\circ}_{i=1}^{L} t^i_{j_i}(\mathbf{r}), \mathbf{w} \right\rangle \qquad \text{eq. 4}$$

where the composition operator is defined

$$\mathop{\circ}_{i=\varnothing,1}^{L} t^l_{j_i}(\mathbf{r}) = \begin{pmatrix} l = 1 \cdots L & t^L_{j_L} \circ t^{L-1}_{j_{L-1}} \cdots \circ t^1_{j_1}(\mathbf{r}) \\ l = \varnothing & \mathbf{r} \end{pmatrix}$$

Let $\mathbf{C}$ be an $L$ dimensional matrix of values of $c(\mathbf{j})$ whose dimensions are $n_1 \ldots n_L$. The problem, then is to find

$$\mathbf{x} = \arg\max c(\mathbf{j}) \qquad \text{eq. 5}$$

The indices $\mathbf{x}$ specify the sequence of transformations that best correspondence between vectors $\mathbf{r}$ and $\mathbf{w}$. The problem is that $\mathbf{C}$ is too large a space to search for $\mathbf{x}$ by conventional means. Instead, a continuous embedding of $\mathbf{C}$ permits a search with resources proportional to the sum of sizes of the dimensions of $\mathbf{C}$ instead of their product.

$C$ is embedded in a superposition dot product space $Q$ defined

$$Q : \mathbb{R}^{\sum_{l=1}^{L} n_l} \to \mathbb{R}^1$$

$$Q(\mathbf{G}) = \left\langle \mathop{\circ}_{l=1}^{m-1} \left( \sum_i g^l_i \cdot t^l_i \right)(\mathbf{r}), \mathop{\circ}_{l=\varnothing,L}^{m+1} \left( \sum_i g^l_i \cdot t'^l_i \right)(\mathbf{w}) \right\rangle \qquad \text{eq. 6}$$

where $\mathbf{G} = \left[ g^m_{x_m} \right]$ $m = 1 \cdots L, x_m = 1 \cdots n_m$ $n_m$ is number of $t$ in layer $m$, $g^m_{x_m} \in [0,1]$, $t'^l_i$ is adjoint of $t^l_i$.

In $Q$ space, the solution to eq. 5 lies along a single axis in the set of axes represented each row of $\mathbf{G}$. That is, $\mathbf{g}^m = <0, \cdots, u_{x_m}, \cdots, 0>$ $u_{x_m} > 0$ which corresponds to the best fitting transformation $t_{x_m}$, where $\mathbf{x}_m$ is the $m^{\text{th}}$ index in $\mathbf{x}$ in eq. 5. This state is reached from an initial state $\mathbf{G} = [1]$ by a process termed *superposition culling* in which the components of grad $Q$ are used to compute a path in steps $\Delta g$,

$$\frac{\partial Q(\mathbf{G})}{g^m_j} = \left\langle t^m_j \mathop{\circ}_{l=\varnothing,1}^{m-1} \left( \sum_i g^l_i \cdot t^l_i \right)(\mathbf{r}), \mathop{\circ}_{l=\varnothing,L}^{m+1} \left( \sum_i g^l_i \cdot t'^l_i \right)(\mathbf{w}) \right\rangle \qquad \text{eq. 7}$$

$$\Delta \mathbf{g}^m = f \left( \frac{\partial Q(\mathbf{G})}{\partial g^m_1}, \cdots, \frac{\partial Q(\mathbf{G})}{\partial g^m_{n_m}} \right) \qquad \text{eq. 8}$$

The function $f$ preserves the maximal component and reduces the others: in neuronal terms, *lateral inhibition*. The resulting path along the surface $Q$ can be thought of as a "high traverse" in contrast to the gradient ascent or descent usual in optimization methods. The price for moving the problem into superposition dot product space is that *collusions* of components of the superpositions can result in better matches for incorrect mappings than for the mappings of the correct solution. If this occurs it is almost always a temporary state early in the convergence. This is a consequence of the *ordering property of superpositions* (OPS) [2,4], which, as applied here, describes the characteristics of the surface $Q$. For example, let three

superpositions $\mathbf{r} = \sum_{i=1}^{n} \mathbf{u}_i$, $\mathbf{s} = \sum_{j=1}^{m} \mathbf{v}_j$ and $\mathbf{s}' = \sum_{k=1}^{m} \mathbf{v}_k$ be formed from three sets of sparse vectors $\mathbf{u}_i \in \mathbf{R}$, $\mathbf{v}_j \in \mathbf{S}$ and $\mathbf{v}_k \in \mathbf{S}'$ where $\mathbf{R} \cap \mathbf{S} = \varnothing$ and $\mathbf{R} \cap \mathbf{S}' = \mathbf{v}_q$. Then the following relationship expresses the OPS:

$$\text{define } P_{correct} = P(\mathbf{r} \bullet \mathbf{s}' > \mathbf{r} \bullet \mathbf{s}), \ P_{incorrect} = P(\mathbf{r} \bullet \mathbf{s}' \le \mathbf{r} \bullet \mathbf{s})$$

$$\text{then } P_{correct} > P_{incorrect} \text{ or } P_{correct} > 0.5$$

$$\text{and as } n,m \rightarrow 1 \quad P_{correct} \rightarrow 1.0$$

Applied to eq. 8, this means that for superpositions composed of vectors which satisfy the distribution properties of sparse, decorrelating encodings[2] (a biologically plausible assumption [6]), the probability of the maximum components of grad $Q$ moving the solution in the correct direction is always greater than 0.5 and increases toward 1.0 as the $\mathbf{G}$ becomes sparser. In other words, the probability of the occurrence of collusion decreases with the decrease in numbers of contributing components in the superposition(s), and/or the decrease in their gating coefficients.

## 3  The map-seeking method and application

A map-seeking circuit (MSC) is composed of several transformation or mapping layers between the input at one end and a memory layer at the other, as seen in Figure 1. The compositional structure is evident in the simplicity of the equations (eqs. 9-12 below) which define a circuit of any dimension. In a multi-layer circuit of $L$ layers plus memory with $n_l$ mappings in layer $l$ the forward path signal for layer $m$ is computed

$$\mathbf{f}^m = \sum_{j=1}^{n_m} g_j^m \cdot t_j^m \left( \mathbf{f}^{m-1} \right) \qquad \text{for } m = 1 \ldots L \qquad\qquad \text{eq. 9}$$

The backward path signal for layer $m$ is computed

$$\mathbf{b}^m = \begin{cases} \sum_{j=1}^{n_l} g_j^m \cdot t_j'^m \left( \mathbf{b}^{m+1} \right) & \text{for } m = 1 \ldots L \\[2ex] \sum_{k} z \left( \mathbf{w}_k \bullet \mathbf{f}^L \right) \cdot \mathbf{w}_k \ \text{ or } \ \sum_{k=1}^{n_w} g_k^m \cdot \mathbf{w}_k \ \text{ or } \ \mathbf{w} & \text{for } m = L+1 \end{cases} \qquad \text{eq. 10}$$

The mapping coefficients $g$ are updated by the recurrence

$$g_i^m := \kappa \left( g_i^m, t_i^m \left( \mathbf{f}^{m-1} \right) \bullet \mathbf{b}^{m+1} \right) \text{ for } m = 1 \ldots L, i = 1 \ldots n_l$$

$$g_i^{L+1} := \kappa \left( g_i^{L+1}, \mathbf{f}^L \bullet \mathbf{w}_k \right) \text{ for } k = 1 \ldots n_w \text{ (optional)} \qquad\qquad \text{eq. 11}$$

where match operator $\mathbf{u} \bullet \mathbf{v} = q$, $q$ is a scalar measure of goodness-of-match between $\mathbf{u}$ and $\mathbf{v}$, and may be non-linear. When $\bullet$ is a dot product, the second argument of $\kappa$ is the same as $\partial Q / g$ in eq. 7. The competition function $\kappa$ is a realization of lateral inhibition function $f$ in eq. 8. It may optionally be applied to the memory layer, as seen in eq. 11.

$$\kappa(g_i, q_i) = \max\left(0, \ g_i - k_1 \cdot \left(1 - \frac{q_i}{\max \mathbf{q}}\right)^{k_2}\right)$$ eq. 12

Thresholds are normally applied to $\mathbf{q}$ and $\mathbf{g}$, below which they are set to zero to speed convergence. In above, $\mathbf{f}^0$ is the input signal, $t_j^m, t_j'^m$ are the $j^{th}$ forward and backward mappings for the $m^{th}$ layer, $\mathbf{w}_k$ is the $k^{th}$ memory pattern, $z(\ )$ is a non-linearity applied to the response of each memory. $\mathbf{g}^m$ is the set of mapping coefficients $g_j^m$ for the $m^{th}$ layer, each of which is associated with mapping $t_j^m$ and is modified over time by the competition function $\kappa(\ )$.

**Recognizing 2D projections of 3D objects under real operating conditions**

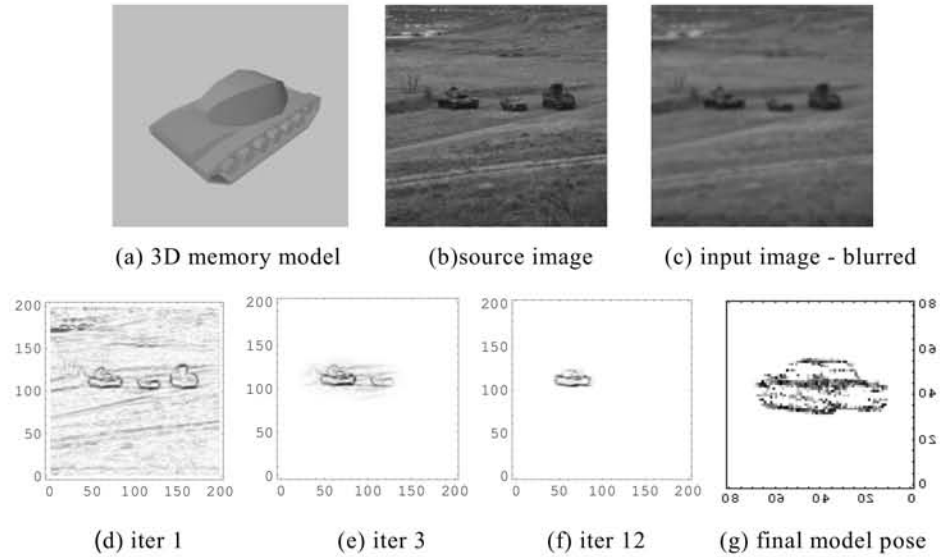

(a) 3D memory model      (b)source image      (c) input image - blurred

(d) iter 1      (e) iter 3      (f) iter 12      (g) final model pose

Figure 2. Recognizing target among distractor vehicles. (a) M60 3D memory model; (b) source image, Fort Carson Data Set; (c) Gaussian blurred input image; (d-f) isolation of target in layer 0, iterations 1, 3, 12; (g) pose determination in final iteration, layer 4 backward - presented left-right mirrored to reflect mirroring determined in layer 3. M-60 model courtesy Colorado State University.

Real world problems of the form expressed in eq. 1 often present objects at distances or in conditions which so limit the resolution that there are no alignable features other than the shape of the object itself, which is sufficiently blurred as to prevent generating reliable edges in a feed-forward manner (e.g. Fig. 2c). In the map-seeking approach, however, the top-down (in biological parlance) inverse-mappings of the 3D model are used to create a set of edge hypotheses on the backward path out of layer 1 into layer 0. In layer 0 these hypotheses are used to gate the input image. As convergence proceeds, the edge hypotheses are reduced to a single edge hypothesis that best fits the grayscale input image. Figure 2 shows this process applied to one of a set of deliberately blurred images from the Fort Carson Imagery Data Set. The MSC used four layers of visual transformations: 14,400 translational, 31 rotational, 41 scaling, 481 3D projection. The MSC had no difficulty distinguishing the location and orientation of the tank, despite distractors

and background clutter: in all tests in the dataset target was correctly located. In effect, once primed with a top-down expectation, attentional behavior is an emergent property of application of the map-seeking method to vision [8].

**Adapting generative models by transformation**

"The direct-matching hypothesis of the interpretation of biological motion] holds that we understand actions when we map the visual representation of the observed action onto our motor representation of the same action." [1]    This mapping, attributed to a neuronal mirror-system for which there is gathering neurobiological evidence (as reviewed in [1]), requires a mechanism for projecting between the visual space and the constrained skeletal joint parameter (kinematic) space to disambiguate the 2D projection of body structure.[4]  Though this problem has been solved to various degrees by other computational methods, a review of which is beyond the scope of this discussion, to the author's knowledge none of these have biological plausibility. The present purpose is to show how simply the problem can be expressed by the generative model interpretation problem introduced in eq. 3 and solve by map-seeking circuits.  An idealized example is the problem of interpreting the shape of a featureless "snake" articulated into any configuration, as appears in Fig. 3.

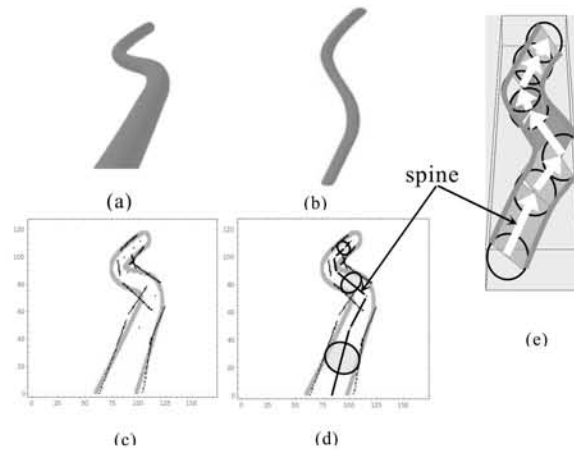

Figure 3. Projection between visual and kinematic spaces with two map-seeking circuits. (a) input view, (b) top view, (c) projection of 3D occluding contours, (d,e) projections of relationship of occluding contours to generating spine.

The solution to this problem involves two coupled map-seeking circuits. The kinematic circuit layers model the multiple degrees of freedom (here two angles, variable length and optionally variable radius from spine to surface) of each of the connected spine segments.  The other circuit determines the visual transformations, as seen in the earlier example.  The surface of the articulated cylinder is mapped from an axial spine. The points where that surface is tangent to the viewpoint vectors define the occluding contours which, projected in 2D, become the object silhouette.  The problem is to find the articulations, segment lengths (and optionally segment diameter) which account for the occluding contour matching the silhouette in the input image. In the MSC solution, the initial state all possible articulations of the snake spine are superposed, and all the occluding contours from a range of viewing angles are projected into 2D.   The latter superposition serves as the backward input to the visual space map-seeking circuit.  Since the snake surfaceis determined by all of the layers of the kinematic circuit, these are projected in

parallel to form the backward (biologically top-down) 2D input to the visual transformation-discovery circuit. A matching operation between the contributors to the 2D occluding contour superposition and the forward transformations of the input image modulates the gain of each mapping in the kinematic circuit via $a_i^m$ in eqs. 13, 14 (modified from eq. 11). In eqs. 13, 14 $K$ indicates kinematic circuit, $V$ indicates visual circuit.

$$\overset{K}{g}_i^m := comp\left( \overset{K}{g}_i^m, \overset{VK}{a}_i^m \cdot \overset{K}{t}_i^m \left( \overset{K}{\mathbf{f}}^{m-1} \right) \bullet \overset{K}{\mathbf{b}}^{m+1} \right) \text{ for } m = 1 \dots \overset{K}{L}, i = 1 \dots \overset{K}{n}_l \qquad \text{eq. 13}$$

$$\overset{VK}{a}_i^m = \left( \overset{V}{\mathbf{f}}^L \right) \bullet t^{3D \to 2D} \circ t^{surface} \circ \overset{K}{t'}_i^m \left( \overset{K}{\mathbf{b}}^{m+1} \right) \qquad \text{eq. 14}$$

The process converges concurrently in both circuits to a solution, as seen in Figure 3. The match of the occluding contours and the input image, Figure 3a, is seen in Figure 3b,c, with its three dimensional structure is clarified in Figure 3d. Figure 3e shows a view of the 3D structure as determined directly from the mapping parameters defining the snake "spine" after convergence.

## 4 Conclusion

The investigations reported here expand the envelope of vision-related problems amenable to a pure transformation-discovery approach implemented by the map-seeking method. The recognition of static 3D models, as seen in Figure 2, and other problems [9] solved by MSC have been well tested with real-world input. Numerous variants of Figure 3 have demonstrated the applicability of MSC to recognizing generative models of high dimensionality, and the principle has recently been applied successfully to real-world domains. Consequently, the research to date does suggest that a single cortical computational mechanism could span a significant range of the brain's visual and kinematic computing.

**References**

[1] G. Rizzolati, L. Fogassi, V. Gallese, Neurophysiological mechanisms underlying the understanding and imitation of action, *Nature Reviews Neuroscience*, 2, 2001, 661-670

[2] D. Arathorn, Map-Seeking: Recognition Under Transformation Using A Superposition Ordering Property. *Electronics Letters* 37(3), 2001 pp164-165

[3] A. Angelucci, B. Levitt, E. Walton, J.M. Hupé, J. Bullier, J. Lund, Circuits for Local and Global Signal Integration in Primary Visual Cortex, *Journal of Neuroscience*, 22(19) , 2002 pp 8633-8646

[4] D. Arathorn, *Map-Seeking Circuits in Visual Cognition*, Palo Alto, Stanford Univ Press, 2002

[5] A. Polsky, B. Mel, J. Schiller, Computational Subunits in Thin Dendrites of Pyramidal Cells, *Nature Neuroscience* 7(6), 2004 pp 621-627

[6] B.A. Olshausen, D.J. Field, Emergence of Simple-Cell Receptive Field Properties by Learning a Sparse Code for Natural Images, *Nature,* 381, 1996 pp607-609

[7] T. Plate, *Holographic Reduced Representation*, CSLI publications, Stanford, California, 2003

[8] D. Arathorn, Memory-driven visual attention: an emergent behavior of map-seeking circuits, in *Neurobiology of Attention*, Eds Itti L, Rees G, Tsotsos J, Academic/Elsevier, 2005

[9] C. Vogel, D. Arathorn, A. Parker, and A. Roorda, "Retinal motion tracking in adaptive optics scanning laser ophthalmoscopy", *Proceedings of OSA Conference on Signal Recovery and Synthesis*, Charlotte NC, June 2005.

## Footnotes

[1] This illustrates that forming a superposition of memories is equivalent to forming superpositions of transformations. The first is a more practical realization, as seen in Figure 1. Though not demonstrated in this paper, the multi-memory architecture has proved robust with 1000 or more memory patterns from real-world datasets.

[2] A restricted case of the superposition ordering property using non-sparse representation is exploited by HRR distributed memory. See [7] for an analysis which is also applicable here.
